# A Delay-Line Based
# Motion Detection Chip

Tim Horiuchi[t]    John Lazzaro[*]    Andrew Moore[t]    Christof Koch[t]

[t]Computation and Neural Systems Program
[*]Department of Computer Science
California Institute of Technology MS 216-76
Pasadena, CA 91125

## Abstract

Inspired by a visual motion detection model for the rabbit retina and by a computational architecture used for early audition in the barn owl, we have designed a chip that employs a correlation model to report the one-dimensional field motion of a scene in real time. Using subthreshold analog VLSI techniques, we have fabricated and successfully tested a 8000 transistor chip using a standard MOSIS process.

## 1. INTRODUCTION

Most proposed short-range intensity-based motion detection schemes fall into two major categories: gradient models and correlation models. In gradient models, computation begins from local image qualities such as spatial gradients and temporal derivatives that can be vulnerable to noise or limited resolution. Correlation models, on the other hand, use a filtered version of the input intensity multiplied with the temporally delayed and filtered version of the intensity at a neighboring

* Present address: John Lazzaro, University of Colorado at Boulder, Campus Box 425, Boulder, Colorado, 80309-0425

receptor. Many biological motion detection systems have been shown to use a correlation model (Grzywacz and Poggio, 1990). To make use of this model, previous artificial systems, that typically look at sampled images of a scene changing in time, have had to cope with the correspondence problem, i.e. the problem of matching features between two images and measuring their shift in position. Whereas traditional digital approaches lend themselves to the measurement of image shift over a fixed time, an analog approach lends itself to the measurement of time over fixed distance. The latter is a local computation that scales to different velocity ranges gracefully without suffering from the problems of extended interconnection.

Inspired by visual motion detection models (Barlow and Levick, 1965) and by a computational architecture found in early audition (Konishi, 1986), we have designed a chip that contains a large array of velocity-tuned "cells" that correlate two events in time, using a delay-line structure. We have fabricated and successfully tested an analog integrated circuit that can can report, in real time, the field motion of a one-dimensional image projected onto the chip. The chip contains 8000 transistors and a linear photoreceptor array with 28 elements.

## 2. SYSTEM ARCHITECTURE

Figure 1 shows the block diagram of the chip. The input to the chip is a real-world image, focused directly onto the silicon via a lens mounted over the chip. The one-dimensional array of on-chip hysteretic photoreceptors (Delbrück and Mead, 1989) receives the light and reports rapid changes in the signal for both large and small changes. Each photoreceptor is connected to a half-wave rectifying neuron circuit (Lazzaro and Mead, 1989) that fires a single pulse of constant voltage amplitude and duration when it receives a quickly rising (but not falling) light-intensity signal.

This rising light intensity signal is interpreted to be a moving edge in the image passing over the photoreceptor. It is this signal that is the "feature" to be correlated. Note that the choice of the rising or falling intensity as a feature, from an algorithmic point of view, is arbitrary. Each neuron circuit is in turn connected to an axon circuit (Mead, 1989) that propagates the pulse down its length. By orienting the axons in alternating directions, as shown in Figure 1, any two adjacent receptors generates pulses that will "race" toward each other and meet at some point along the axon. Correlators between the axons detect when pulses pass each other, indicating the detection of a specific time difference. The width of the pulses in the axon circuits is adjustable and determines the detectable velocity range. From the summing of "votes" for different velocities by correlators across the entire chip, a winner-take-all circuit (Lazzaro et al., 1989) determines the velocity.

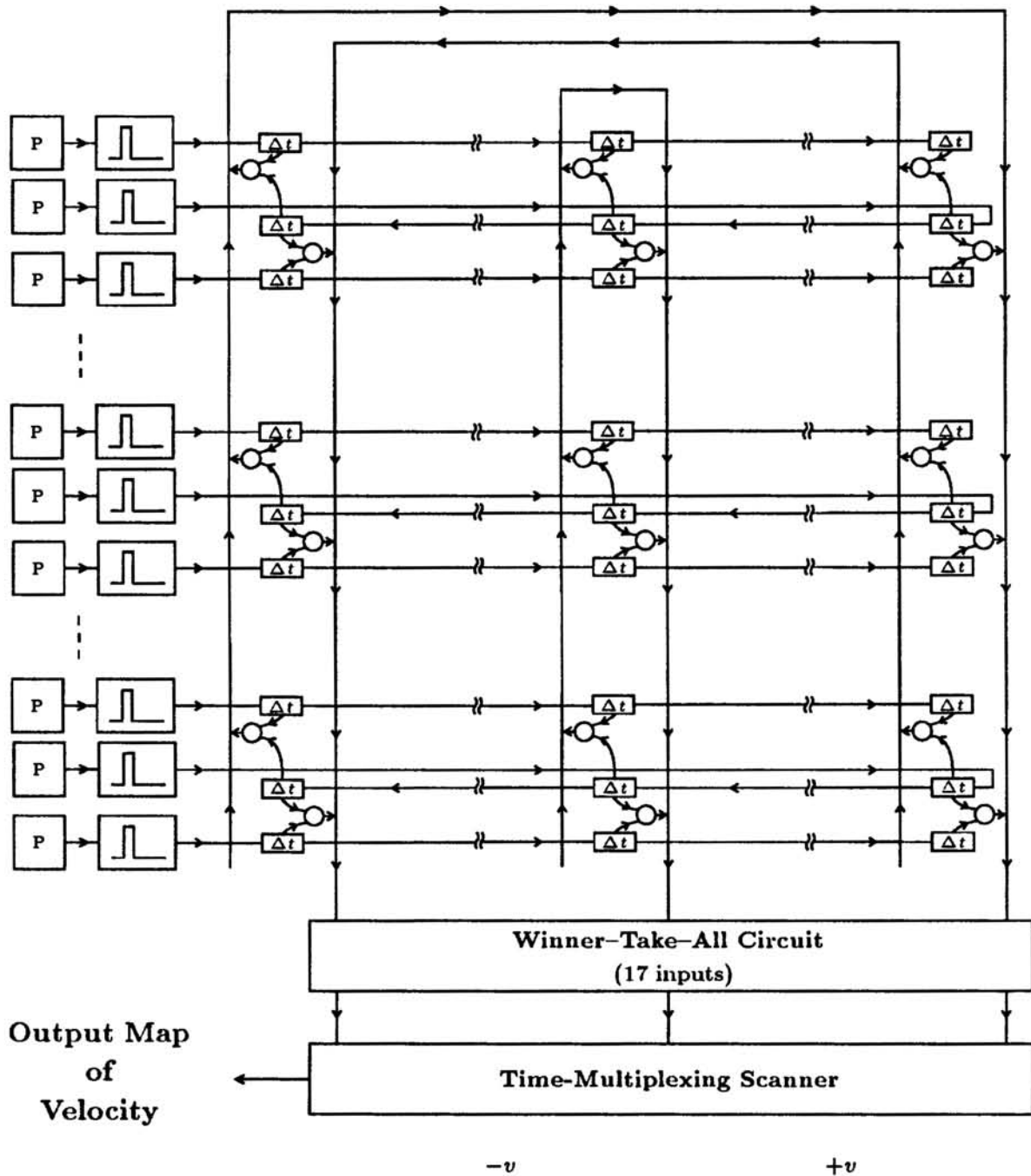

Figure 1. Block diagram of the chip, showing information flow from the photoreceptors (P), to the time-multiplexed winner-take-all output. Rising light signals are converted to pulses that propagate down the axons. Correlators are drawn as circles and axons are piecewise denoted by $\Delta t$ boxes. See the text for explanation.

# 3. SYSTEM OPERATION AND RESULTS

## 3.1 READING BETWEEN THE LINES

The basic signal quantity that we are measuring is the time a "feature" takes to travel from one photoreceptor to one of its neighbors. By placing two delay lines in parallel that propagate signals in opposing directions, a temporal difference in signal start times from opposite ends will manifest itself as a difference in the location where the two signals will meet. Between the axons, correlation units perform a logical AND with the axon signals on both sides. If pulses start down adjacent axons with zero difference in start times (i.e. infinite velocity), they will meet in the center and activate a correlator in the center of the axon. If the time difference is small (i.e. the velocity is large), correlations occur near the center. As the time difference increases, correlations occur further out toward the edges. The two halves of the axon with respect to the center represent different directions of motion. When a single stimulus (e.g. a step edge) is passed over the length of the photoreceptor array with a constant velocity, a specific subset of correlators will be activated that all represent the same velocity. A current summing line is connected to each of these correlators and is passed to a winner-take-all circuit. The winner of the winner-take-all computation corresponds to the line that is receiving the largest number of correlation inputs. The output of the winner-take-all is scanned off the chip using an external input clock. Because the frequency of correlation affects the confidence of the data, scenes that are denser in edges provide more confident data as well as a quicker response.

## 3.2 SINGLE VS. BURSTING MODE

Until now, the circuit described uses a single pulse to indicate a passing edge. Due to the statistical nature of this system, a large number of samples are needed to make a confident statement of the detected time difference, or velocity. By externally increasing the amplitude of the signal passed to the neuron during each event, the neuron can fire multiple pulses in quick succession. With an increased number of pulses travelling down the axon, the number of correlations increase, but with a decrease in accuracy, due to the multiple incorrect correlations. The incorrect correlations are not random, however, but occur closely around the correct velocity. The end result is a net decrease in resolution in order to achieve increased confidence in the final data.

## 3.3 VELOCITY RANGE

The chip output is the measured time difference of two events in multiples of $\tau$, the time-constant of a single axon section. The time difference (measured in seconds/pixel) is translated into velocity, by the equation $V = 1/\Delta t$, where $V$ is velocity in pixels/sec and $\Delta t$ can be positive or negative. Thus the linear measurement of time difference gives a non-linear velocity interpretation with the highest resolution

at the slower speeds. At the slower speeds, however, we tend to have decreased confidence in the data due to the relatively smaller correlation frequency. This is expected to be less troublesome as larger photoreceptor arrays are used. The variable resolution in the computation is often an acceptable feature for control of robotic motion systems since high velocity motions are often ballistic or at least coarse, whereas fine control is needed at lower velocities.

## 3.4 PERFORMANCE

We have fabricated the circuit shown in Figure 1 using a double polysilicon $2\mu m$ process in the MOSIS Tiny Chip die. The chip has 17 velocity channels, and an input array of 28 photoreceptors. The voltages from the winner-take-all circuit are scanned out sequentially by on-chip scanners, the only clocked circuitry on the chip.

In testing the chip, gratings of varying spatial frequencies and natural images from newspaper photos and advertisements were mounted on a rotating drum in front of the lens. Although the most stable data was collected using the gratings, both images sources provided satisfactory data. Figure 2 shows oscilloscope traces of scanned winner-take-all channels for twelve different negative and positive velocities within a specific velocity range setting. The values to the right indicate the approximate center of the velocity range. Figure 3(a) shows the winning time interval channel vs. actual time delay. The response is linear as expected. Figure 3(b) shows the data from Figure 3(a) converted to the interpreted velocity channel vs. velocity. The horizontal bars indicate the range of velocity inside of which each channel responds. As described above, at the lower velocities, correlations occur at a lower rate, thus some of the lowest velocity channels do not respond. By increasing the number of parallel photoreceptor channels, it is expected that this situation will improve. The circuit, currently with only eight velocity channels per direction, is able to reliably measure, over different settings, velocities from 2.9 pixels/sec up to 50 pixels/sec.

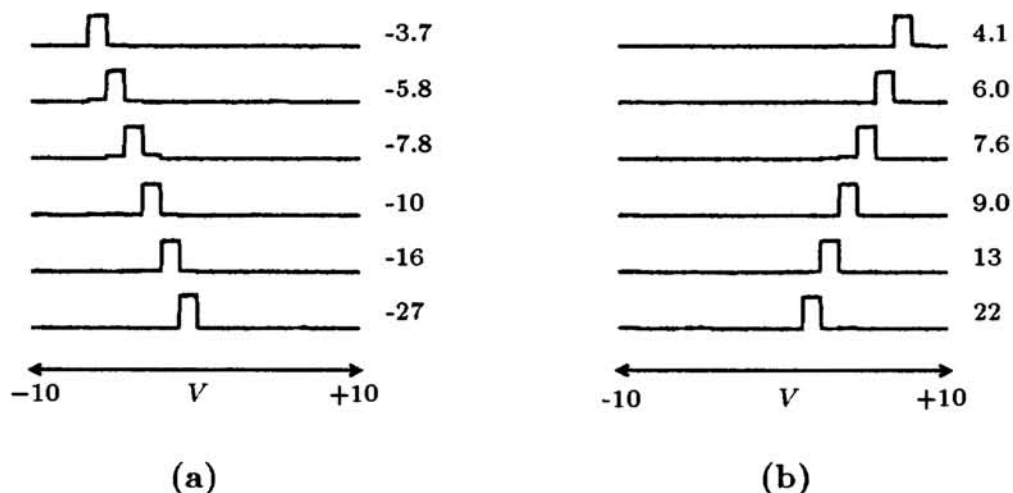

Figure 2. Winner-take-all oscilloscope traces for twelve positive (a) and negative (b) velocities. Trace labels represent the approximate center of the velocity range.

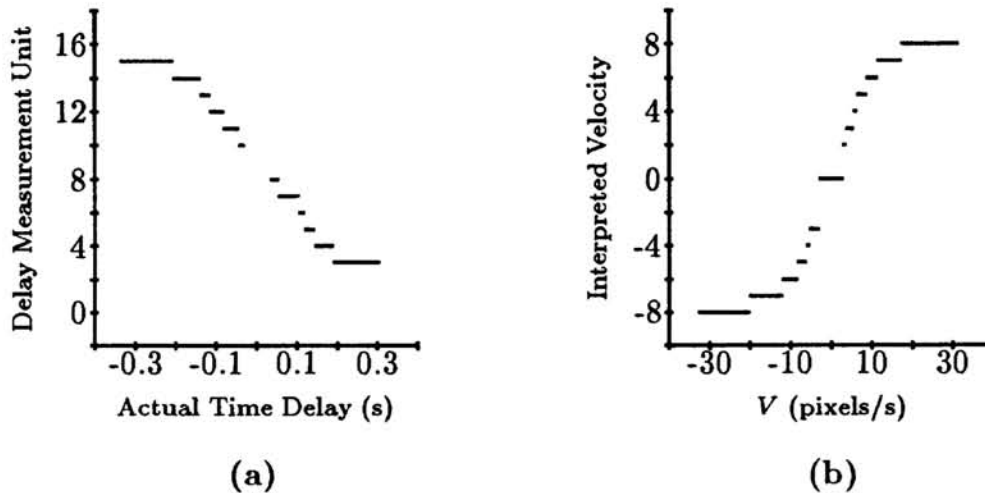

Figure 3. (a) Plot of winning time interval channel vs. actual time delay. (b) Plot of interpreted velocity channel vs. velocity (same data as in (a)).

An interesting feature of our model that also manifests itself in the visual system of the fly (Buchner 1984) is spatial aliasing, leading in the worst case to motion reversal. Spatial aliasing is due to the discrete sampling provided by photoreceptor spacing. At spatial frequencies higher than the Nyquist limit, a second stimulus can enter the neighboring axon before the first stimulus has exited, causing a sudden change in the sign of the velocity.

# 4 CONCLUSION

A correlation-based model for motion detection has been successfully demonstrated in subthreshold analog VLSI. The chip has shown the ability to successfully detect relatively low velocities; the slowest speed detected was 2.9 pixels/sec. and shows promise for use in different settings where other motion detection strategies have difficulty. The chip responds very well to low-light stimulus and its output is robust against changes in contrast. This is due to the high temporal derivative sensitivity of the hysteretic photoreceptor to both large and small changes. Interestingly, the statistical nature of the computation allows the system to perform successfully in noise as well as to produce a level of confidence measure. In addition, the nature of the velocity computation provides the highest resolution at the slower speeds and may be considered as an effective way to expand the detectable velocity range.

### Acknowledgements

We thank Carver Mead for providing laboratory resources for the design, fabrication, and initial testing of this chip. We thank Rockwell International and the Hughes Aircraft Corporation for financial support of VLSI research in Christof Koch's laboratory, and we thank the System Development Foundation and the Office Naval Research for financial support of VLSI research in Carver Mead's laboratory. We thank Hewlett-Packard for computing support and the Defense Advanced Research

Projects Agency and the MOS Implementation Service (MOSIS) for chip fabrication.

## References

Barlow, H.B. and Levick, W.R. (1965) The mechanism of directionally sensitive units in rabbit's retina. *J. Physiol.* **178**: 477-504.

Buchner, E. (1984). Behavioural Analysis of Spatial Vision in Insects. In Ali, M. A. (ed) *Photoreception and Vision in Invertebrates.* New York: Plenum Press, pp. 561-621.

Delbrück, T. and Mead, C. (1989) An Electronic Photoreceptor Sensitive to Small Changes in Intensity. In Touretzky (ed), *Neural Information Processing Systems 1.* San Mateo, CA: Morgan Kaufmann Publishers, pp. 720-727.

Grzywacz, N. and Poggio, T. (1990). Computation of Motion by Real Neurons. In Zornetzer (ed), *An Introduction to Neural and Electronic Networks.* New York: Academic Press, pp. 379-401.

Konishi, M. (1986). Centrally synthesized maps of sensory space. *Trends in Neuroscience* **4**: 163-168.

Lazzaro, J. and Mead, C. (1989). Circuit models of sensory transduction in the cochlea. In Mead, C. and Ismail, M. (eds), *Analog VLSI Implementations of Neural Networks.* Norwell, MA: Kluwer Academic Publishers, pp. 85-101.

Lazzaro, J., Ryckebusch, S., Mahowald, M. A., and Mead, C. (1988). Winner-take-all networks of O(n) complexity. In Tourestzky, D. (ed), *Advances in Neural Information Processing Systems 1.* San Mateo, CA: Morgan Kaufmann Publishers, pp. 703-711.

Mead., C. (1989) *Analog VLSI and Neural Systems.* Reading, MA: Addison-Wesley, pp. 193-203.


